# Learning Multiple Tasks using Manifold Regularization

**Arvind Agarwal**[*]      **Hal Daumé III**[*]
Department of Computer Science
University of Maryland
College Park, MD 20740
arvinda@cs.umd.edu
hal@umiacs.umd.edu

**Samuel Gerber**
Scientific Computing and Imaging Institute
University of Utah
Salt Lake City, Utah 84112
sgerber@cs.utah.edu

## Abstract

We present a novel method for multitask learning (MTL) based on *manifold regularization*: assume that all task parameters lie on a manifold. This is the generalization of a common assumption made in the existing literature: task parameters share a common *linear* subspace. One proposed method uses the projection distance from the manifold to regularize the task parameters. The manifold structure and the task parameters are learned using an alternating optimization framework. When the manifold structure is fixed, our method decomposes across tasks which can be learnt independently. An approximation of the manifold regularization scheme is presented that preserves the convexity of the single task learning problem, and makes the proposed MTL framework efficient and easy to implement. We show the efficacy of our method on several datasets.

## 1   Introduction

Recently, it has been shown that learning multiple tasks together helps learning [8, 19, 9] when the tasks are *related*, and one is able to use an appropriate notion of task relatedness. There are many ways by which one can enforce the relatedness of the tasks. One way to do so is to assume that two tasks are related if their parameters are "close". This notion of relatedness is usually incorporated in the form of a regularizer [4, 16, 13] or a prior [15, 22, 21].

In this work we present a novel approach for multitask learning (MTL) that considers a notion of relatedness based on ideas from *manifold regularization*[1]. Our approach is based on the assumption that the parameters of related tasks can not vary arbitrarily but rather lie on a low dimensional manifold. A similar idea underlies the standard manifold learning problems: the data does not change arbitrarily, but instead follows a manifold structure. Our assumption is also a generalization of the assumption made in [1] which assumes that all tasks share a *linear subspace*, and a learning framework consists of learning this linear subspace and task parameters simultaneously. We remove the linear constraint from this problem, and assume that the tasks instead share a *non-linear* subspace.

In our proposed approach we learn the task parameters and the task-manifold alternatively, learning one while keeping the other fixed, similar to [4]. First, we learn all task parameters using a single task learning (STL) method, and then use these task parameters to learn the initial task manifold. The task-manifold is then used to relearn the task parameters using manifold regularization. Learning of manifold and task parameters is repeated until convergence. We emphasize that when we learn the task parameters (keeping the manifold structure fixed), the MTL framework decomposes across the

---

[*]This work was done at School of Computing, University of Utah, Salt Lake City, Utah

[1]It is not to be confused with the manifold regularization presented in [7]. We use the projection distance for regularization while Belkin et.al. use the graph structure (graph Laplacian).

tasks, which can be learned independently using standard method such as SVMs. Note that unlike most manifold learning algorithms, our framework learns an *explicit* representation of the manifold and naturally extends to new tasks. Whenever a new task arrives, one can simply use the existing manifold to learn the parameters of the new task. For a new task, our MTL model is very efficient as it does not require relearning all tasks.

As shown later in the examples, our method is simple, and can be implemented with only a small change to the existing STL algorithms. Given a black box for manifold learning, STL algorithms can be adapted to the proposed MTL setting. To make the proposed framework even simpler, we provide an approximation which preserves the convexity of the STL problem. We emphasize that this approximation works very well in practice. All the experimental results used this approximation.

## 2 Related Work

In MTL, task relatedness is a fundamental question and models differ in the ways they answer this question. Like our method, most of the existing methods first assume a structure that defines the task relatedness, and then incorporate this structure in the MTL framework in the form of a regularizer [4, 16, 13].

One plausible approach is to assume that all task parameters lie in a subspace [1]. The tasks are learned by forcing the parameters to lie in a common linear subspace therefore exploiting the assumed relatedness in the model. Argyriou et.al. [4] later generalized this work by using a function $F$ to model the shared structure. In this work, the relatedness structure is forced by applying a function $F$ on a covariance matrix $D$ which yields a regularization of the form $tr(F(D)WW^T)$ on the parameters $W$. Here, the function $F$ can model different kind of relatedness structures among tasks including the linear subspace structure [1]. Given a function $F$, this framework learns both, the relatedness matrix $D$ and the task parameters $W$. One of the limitations of this approach is the dependency on $F$ which has to be provided externally. In an informal way, $F$ introduces the non-linearity and it is not clear as what the right choice of $F$ is. Our framework generalizes the linear framework by introducing the nonlinearity through the manifold structure learned automatically from the data, and thus avoids the need of any external function. Argyriou et. al. extend their work [4] in [2, 3] where non-linearity is introduced by considering a kernel function on the input data, and then learning the linear subspace in the Hilbert space. This method in spirit is very similar to our method except that we learn an explicit manifold therefore our method is naturally extensible to new tasks.

Another work that models the task relatedness in the form of proximity of the parameters is [16] which assumes that task parameters $w_t$ for each task is close to some common task $w_0$ with some variance $v_t$. These $v_t$ and $w_0$ are learned by minimizing the *Euclidean* norm which is again equivalent to working in the linear space. This idea is later generalized by [13], where tasks are clustered, and regularized with respect to the cluster they belong to. The task parameters are learned under this cluster assumption by minimizing a combination of different penalty functions.

There is another line of work [10], where task relatedness is modeled in term of a matrix $B$ which needs to be provided externally. There is also a large body of work on multitask learning that find the shared structure in the tasks using Bayesian inference [23, 24, 9], which in spirit, is similar to the above approaches, but done in a Bayesian way. It is to be noted that all of the above methods either work in a linear setting or require external function/matrix to enforce the nonlinearity. In our method, we work in the non-linear setting without using any external function.

## 3 Multitask Learning using Manifold

In this section we describe the proposed MTL framework. As mentioned earlier, our framework assumes that the tasks parameters lie on a manifold which is a step further to the assumption made in [1] i.e., the task parameters lie on a linear subspace or share a common set of features. Similar to the linear subspace algorithm [1] that learns the task parameters (and the shared subspace) by regularizing the STL framework with the orthogonal projections of the task parameters onto the subspace, we propose to learn the task parameters (and non-linear subspace i.e., *task-manifold*) by

regularizing the STL with the projection distance of the task parameters from this task-manifold (see Figure 1).

We begin with some notations. Let $T$ be the total number of tasks, and for each task $t$, let $X_t = \{x_1, \ldots x_{n_t}\}$ be the set of examples and $Y_t = \{y_1, \ldots y_{n_t}\}$ be the corresponding labels. Each example $x_i \in \mathbb{R}^d$ is a $d$ dimensional vector, and $y_i$ is a label; $y_i \in \{+1, -1\}$ in case of a classification problem, and a real value $y_i \in \mathbb{R}$ in case of regression problem. $n_t$ is the number of examples in task $t$. For the simplicity of the notations, we assume that all tasks have the same number of examples i.e. $n_1 = \ldots = n_T = n$, though in practice they may vary. Now for each task $t$, let $\theta_t$ be the parameter vector, referred as the *task parameter*.

Given example-label pairs set $(X_t, Y_t)$ for task $t$, a learning problem would be to find a function $f_t$ that for any future example $x$, predicts the correct value of $y$ i.e. $y = f_t(x)$. A standard way to learn this function is to minimize the loss between the value predicted by the function and the true value. Let $\mathcal{L}$ be such a loss function. Let $k$ be a kernel defined on the input examples $k : \mathbb{R}^d \times \mathbb{R}^d \to \mathbb{R}$ and $\mathcal{H}_k$ be the reproducing kernel Hilbert space (RKHS) associated with the kernel $k$. Restricting $f_t$ to the functions in the RKHS and denoting it by $f(x, \theta_t) = \langle \theta_t, \phi(x) \rangle$, single task learning solves the following optimization problem:

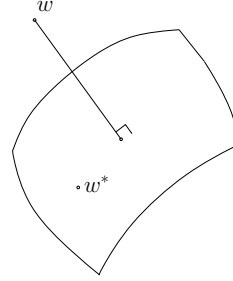

$$\theta_t^* = \underset{\theta_t}{\arg\min} \sum_{x \in X_t} \mathcal{L}(f(x; \theta_t), y) + \lambda \, ||f_t||^2_{\mathcal{H}_k}, \quad (1)$$

Figure 1: Projection of the estimated parameters $w$ of the task in hand on the manifold learned from all tasks parameters. $w^*$ is the optimal parameter.

here $\lambda$ is a regularization parameter. Note that the kernel is assumed to be common for all tasks hence does not have the subscript $t$. This is equivalent to saying that all tasks belong to the same RKHS.

Now one can extend the above STL framework to the multitask setting. In MTL, tasks are related, this notion of relatedness is incorporated through a regularizer. Let $u$ be such regularizer, then MTL solves:

$$(\theta_1^*, \ldots \theta_T^*) = \underset{(\theta_1, \ldots \theta_T)}{\arg\min} \sum_{t=1}^{T} \Big( \sum_{x \in X_t} \mathcal{L}(f(x; \theta_t), y) + \lambda \, ||f_t||^2_{\mathcal{H}_k} \Big) + \gamma u(\theta_1 \ldots \theta_T), \quad (2)$$

where $\gamma$ is a trade off parameter similar to $\lambda$ that trades off the amount of MTL regularization. As mentioned in Section 2, there are many ways in which this regularizer can be implemented. For example, for the assumption that the task parameters are close to a common task $\theta_0$, regularizer would just be $||\theta_t - \theta_0||^2$. In our approach, we split the regularizer $u(\theta_1, \ldots, \theta_T)$ into $T$ different regularizers $u(\theta_t, \mathcal{M})$ such that $u(\theta_t, \mathcal{M})$ regularizes the parameter of task $t$ while considering the effect of other tasks through the manifold $\mathcal{M}$. The optimization problem under such regularizer can be written as:

$$(\theta_1^*, \ldots \theta_T^*) = \underset{(\theta_1, \ldots \theta_T), \mathcal{M}}{\arg\min} \sum_{t=1}^{T} \Big( \sum_{x \in X_t} \mathcal{L}(f(x; \theta_t), y) + \lambda \, ||f_t||^2_{\mathcal{H}_k} + \gamma u(\theta_t, \mathcal{M}) \Big). \quad (3)$$

Note that optimization is now performed over both task parameters and the manifold. If manifold structure $\mathcal{M}$ is fixed then the above optimization problem decomposes into $T$ independent optimization problems. In our approach, the regularizer depends on the structure of the manifold constructed from the task parameters $\{\theta_1, \ldots \theta_T\}$. Let $\mathcal{M}$ be such manifold, and $\mathcal{P}_{\mathcal{M}}(\theta_t)$ be the projection distance of $\theta_t$ from the manifold. Now one can use this projection distance as a regularizer $u(\theta_t, \mathcal{M})$ in the cost function since all task parameters are assumed to lie on the task manifold $\mathcal{M}$. The cost function is now given by:

$$\mathcal{C}_{\mathcal{P}} = \sum_{t=1}^{T} \Big( \sum_{x \in X_t} \mathcal{L}(f(x; \theta_t), y) + \lambda \, ||f_t||^2_{\mathcal{H}_k} + \gamma \mathcal{P}_{\mathcal{M}}(\theta_t) \Big). \quad (4)$$

Since the manifold structure is not known, the cost function (4) needs to be optimized simultaneously for the task parameters $(\theta_1 \ldots \theta_T)$ and for the task-manifold $\mathcal{M}$. Optimizing for $\theta$ and $\mathcal{M}$ jointly is a hard optimization problem, therefore we resort to the alternating optimization. We first

fix the task parameters and learn the manifold. Next, we fix the manifold $\mathcal{M}$, and learn the task parameters by minimizing (4). In order to minimize (4) for the task parameters, we need an expression for $\mathcal{P}_\mathcal{M}$ i.e. an expression for computing the projection distance of task parameters from the manifold. More precisely, we only need the gradient of $\mathcal{P}_\mathcal{M}$ not the function itself since we will solve this problem using gradient descent.

## 3.1 Manifold Regularization

Our approach relies heavily on the capability to learn a manifold, and to be able to compute the gradient of the projection distances onto the manifold. Much recent work in manifold learning focused on uncovering low dimensional representation [18, 6, 17, 20] of the data. These approaches do not provide the tools crucial to this work i.e., the gradient of the projection distance. Recent work [11] addresses this issues and proposes a manifold learning algorithm, based on the idea of principal surfaces [12]. It *explicitly* represents the manifold in the ambient space as a parametric surface which can be used to compute the projection distance and its gradient.

For the sake of completeness, we briefly describe this method (for details refer [11]). The method is based on minimizing the expected reconstruction error $E[g(h(\theta)) - \theta]$ of the task parameter $\theta$ onto the manifold $\mathcal{M}$. Here $h$ is the mapping from the manifold to the lower dimensional Euclidean space and $g$ is the mapping from the lower dimensional Euclidean space to the manifold. Thus, the composition $g \circ h$ maps a point belonging to manifold to the manifold, using the mapping to the Euclidean space as an intermediate step. Note that $\theta$ and $g(h(\theta))$ are usually not the same. These mappings $g$ and $h$ can be formulated in terms of kernel regressions over the data points:

$$h(\theta) = \sum_{j=1}^{T} \frac{K_\theta(\theta - \theta_j)}{\sum_{l=1}^{T} K_\theta(\theta - \theta_l)} z_j \tag{5}$$

with $K_\theta$ a kernel function and $z_j$ a set of parameters to be estimated in the manifold learning process. Similarly

$$g(r) = \sum_{j=1}^{T} \frac{K_r(r - h(\theta_j))}{\sum_{l=1}^{T} K_r(r - h(\theta_l))} \theta_j \tag{6}$$

again with $K_r$ a kernel function.

Note that in the limit, the kernel regression converges to the conditional expectation $g(r) = E[(\theta_1, \ldots, \theta_T)|r]$ where expectation is taken with respect to probability distribution $p(\theta)$, parameters are assumed to be sampled from. If $h$ is an orthogonal projection, this yields a principal surface [12], i.e informally $g$ passes through the middle of the density. In [11] it is shown that in the limit, as the number of samples to learn from increases, $h$ indeed yields an orthogonal projection onto $g$. Under this orthogonal projection, the estimation of the parameters $z_i$, i.e. the manifold learning, can be done through gradient descent on the sample mean of the projection distance $\frac{1}{T} \sum_{i=1}^{T} g(h(\theta_i)) - \theta_i$ using a global manifold learning approach for initialization. Once $h$ is estimated, the projection distance is immediate by

$$\mathcal{P}_\mathcal{M} = \|\theta - g(h(\theta))\|^2 = \|\theta - \theta^\mathcal{M}\|^2 \tag{7}$$

For the optimization of (4) we need the gradient of the projection distance which is

$$\frac{d\mathcal{P}_\mathcal{M}(\theta)}{d\theta} = 2(g(h(\theta)) - \theta) \frac{dg(r)}{dr}\Big|_{r=h(\theta)} \frac{dh(\theta)}{d\theta}. \tag{8}$$

The projection distance for a single task parameters is $O(n)$ due to the definition of $h$ and $g$ as kernel regressions which show up in the projection distance gradient in $\frac{dg(r)}{dr}\Big|_{r=h(\theta)}$ and $\frac{dh(\theta)}{d\theta}$. This is fairly expensive therefore we propose an approximation, justified by the convergence to an orthogonal projection of $h$, to the exact projection gradient. For an orthogonal projection the term $\frac{dg(r)}{dr}\Big|_{r=h(\theta)} \frac{dh(\theta)}{d\theta}$ vanishes ($\frac{dh(\theta)}{d\theta}$ is orthogonal to the tangent plane $\frac{dg(r)}{dr}\Big|_{r=h(\theta)}$ of the projected point) and the gradient simplifies to

$$\frac{d\mathcal{P}_\mathcal{M}(\theta)}{d\theta} = 2(g(h(\theta)) - \theta), \tag{9}$$

which is exactly the gradient of (7) assuming that the projection of $\theta$ onto the manifold is fixed. A further advantage of this approximation , besides a computational speedup, is that no non-convexities are introduced due to the regularization.

**Algorithm 1** MTL using Manifold Regularization

---

**Input:** $\{x_i, y_i\}_{i=1}^n$ for $t = 1 \ldots T$.
**Output:** $\theta_1, \ldots \theta_T$.
**Initialize:** Learn $\theta_1, \ldots \theta_T$ independently.
Learn the task-manifold using $\theta_1, \ldots \theta_T$.
**while** it $<$ numIter **do**
   **for** $t = 1$ to $T$ **do**
      Learn $\theta_t$ using (4) with (7) or (10).
   **end for**
   Relearn the task-manifold using $\theta_1, \ldots \theta_T$.
**end while**

---

The proposed manifold regularization approximation allows to use any STL method without much change in the optimization of the STL problem. The proposed method for MTL pipelines manifold learning with the STL. Using (7) one can write the (4) as:

$$\mathcal{C}_{\mathcal{P}} = \sum_{t=1}^{T} \Big( \sum_{x \in X_t} \mathcal{L}(f(x; \theta_t), y) + \lambda \|\theta_t\|^2 + \gamma \left\|\theta_t - \tilde{\theta}_t^{\mathcal{M}}\right\|^2 \Big) \tag{10}$$

here $\tilde{\theta}_t^{\mathcal{M}}$ is the fixed projection of the $\theta$ on the manifold. Note that in the proposed approximation of the above expression, $\tilde{\theta}_t^{\mathcal{M}}$ is fixed while computing the gradient i.e., one does not have to worry about moving the projection of the point on the manifold during the gradient step. Although in the following example, we will solve (10) for linear kernel, extension for the non-linear kernels is straightforward under the proposed approximation. This approximation allows one to treat the manifold regularizer similar to the RKHS regularizer $\|\theta_t\|^2$ and solve the generalized learning problem (4) with non-linear kernels. Note that $\|\theta_t - \tilde{\theta}_t^{\mathcal{M}}\|^2$ is a monotonic function of $\theta$ so it does not violate the representer theorem.

### 3.2 Example: Linear Regression

In this section, we solve the optimization problem (4) for the linear regression model. This is the model we have used in all of our experiments. In the learning framework (4), the loss function is $\mathcal{L}(x, y, w_t) = (y - \langle w_t, x \rangle)^2$ with linear kernel $k(x, y) = \langle x, y \rangle$. We have changed the notations for parameters from $\theta$ to $w$ to differentiate the linear regression from the general framework. The cost function for linear regression can now be written as:

$$\mathcal{C}_{\mathcal{P}} = \sum_{t=1}^{T} \Big( \sum_{x \in X_t} (y - \langle w_t, x \rangle)^2 + \frac{\lambda}{2} \|w_t\|^2 + \gamma \mathcal{P}_{\mathcal{M}}(w_t) \Big) \tag{11}$$

This cost function may be convex or non-convex depending upon the manifold terms $\mathcal{P}_{\mathcal{M}}(w_t)$. The first two terms are convex. If one uses the approximation (10), this problem becomes convex and has the form similar to STL. The solution under this approximation is given by:

$$w_t = \big((\lambda + \gamma)I + \langle X_t, X_t^T \rangle\big)^{-1} \big(X_t Y_t^T + \gamma \tilde{w}_t^{\mathcal{M}}\big) \tag{12}$$

where $I$ is a $d \times d$ identity matrix, $X_t$ is a $d \times n$ example matrix, and $Y_t$ is a row vector of corresponding labels. $\tilde{w}_t^{\mathcal{M}}$ is the orthogonal projection of $w$ on the manifold.

### 3.3 Algorithm Description

The algorithm for MTL with manifold regularization is straightforward and shown in Algorithm 1. The algorithm begins with the STL setting i.e., each task parameter is learned independently. These learned task parameters are then used to estimate the task-manifold. Keeping the manifold structure fixed, we relearn all task parameters using manifold regularization. Equation (9) is used to compute the gradient of the projection distance used in relearning the parameters. This step gives us the explicit representation of the projection in the case of a linear kernel while a set of weights in the case of a non-linear kernel. Current code available for computing the projection [11] only handles points in the Euclidean space (RKHS with linear kernel), not in a general RKHS, though in theory, it is possible to extend the current code to general RKHS. Once the parameters for all tasks are learned, the manifold is re-estimated based on the updated task parameters. This process is repeated for a fixed number of iterations (in our experiments we use 5 iterations).

## 4 Experiments

In this section, we consider the regression task and show the experimental results of our method. We evaluate our method on both synthetic and real datasets.

### 4.1 Synthetic Dataset

First, we evaluate our method on a *synthetic* data. This data is generated from the task parameters sampled from a known manifold (*swiss roll*). The data is generated by first sampling the points from the 3-dimensional *swiss roll*, and then using these points as the task parameters to generate the examples using the linear regression model. We sample 100 tasks, and for each task we generate 2 examples. The number of examples per task is kept low for two reasons. First, the task at hand (this is linear) is a relatively *easy* task and more number of examples give a nearly perfect regression model with the STL method itself, leaving almost no room for improvement. Second, MTL in the real world makes sense only when the number of examples per task is low. In all of our experiments, we compare our approach with the approach presented in [4] for two reasons. First, this is the approach most closely related to our approach (this makes linear assumption while we make the non-linear assumption), and second, code is available online[2] .

In all our experiments we report the *root mean square error* (RMSE) [4]. For a set of 100 tasks, taskwise results for the synthetic data is shown in Figure 2(a). In this figure, the $x$-axis represents the RMSE of the STL model while the $y$-axis is the RMSE of the MTL model. Figure 2(a) shows the performance of the MTL model relative to the STL model. Each point $(x, y)$ in the figure represents the (STL,MTL) pair. Blue dots denote the MTL performance of our method while green crosses denote the performance of the baseline method [4]. The red line denote the points where MTL and STL performed equally. Any point above the red line shows that the RMSE of MTL is higher (bad case) while points below denote that RMSE of MTL is lower (good case). It is clear from Figure 2(a) that our method is able to use the manifold information therefore outperform both STL and MTL-baseline methods. We improve the performance of almost all tasks with respect to STL, while MTL-baseline improves the performance of only few tasks. Note the mean performance *improvement* (reduction in RMSE i.e. RMSE of (STL-MTL)) of all tasks in our method and in the baseline-MTL. We get an improvement of $+0.0131$ while baseline has the negative performance improvement of $-0.0204$. For the statistical significance, reported numbers are averaged over 10 runs. Hyperparameters of both models (baseline and ours ($\lambda$ and $\gamma$)) were tuned on a small dataset chosen randomly.

### 4.2 Real Regression Dataset

We now evaluate our method on two real datasets *school dataset* and *computer survey dataset* [14], the same datasets as used in the baseline model [4]. Moreover they have also been used in previous MTL studies, for example, school dataset in [5, 10] and computer dataset in [14].

**Computer**   This dataset is a survey of 190 students who rated the likelihood of purchasing one of 20 different personal computers. Here students correspond to the tasks and computers correspond to the examples. Each student rated all of the 20 computers on a scale of 0-10, therefore giving 20 labeled examples per task. Each computer (input example) is represented by 13 different computer characteristics (RAM, cache, CPU, price etc.). Training and test sets were obtained by splitting the dataset into 75% and 25%, thus giving 15 examples for training and 5 examples for testing.

**School**   This dataset [3] is from the Inner London Education Authority and consists of the examination scores of 15362 students from 139 schools in London. Here, each school corresponds to a task, thus a total of 139 tasks. The input consists of the year of the examination, 4 school-specific and 3 student-specific attributes. Following [5, 4], each categorical feature is replaced with binary

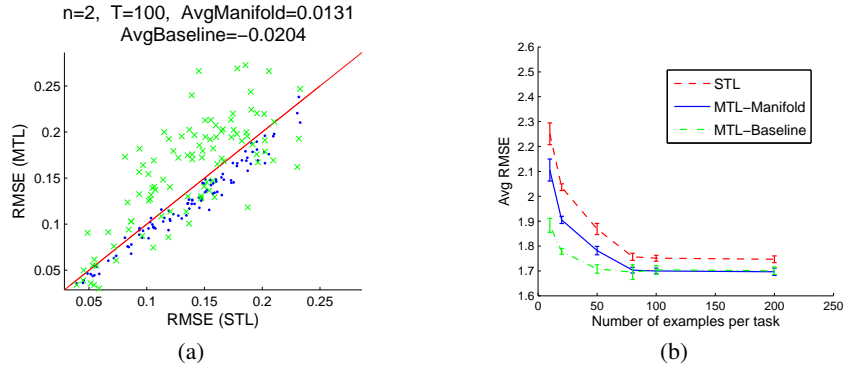

(a)  (b)

Figure 2: Taskwise performance on the synthetic dataset. The red line marks where STL and MTL perform equally. Any points above it represent the tasks whose RMSE increased through the MTL framework while those below showed performance improvement (reduced RMSE). Green crosses are the baseline method and blue dots are the manifold method. $Avg\{Manifold,Baseline\}$ in the title is the mean performance improvement of all tasks over STL. (b) Average RMSE vs number of examples for school dataset

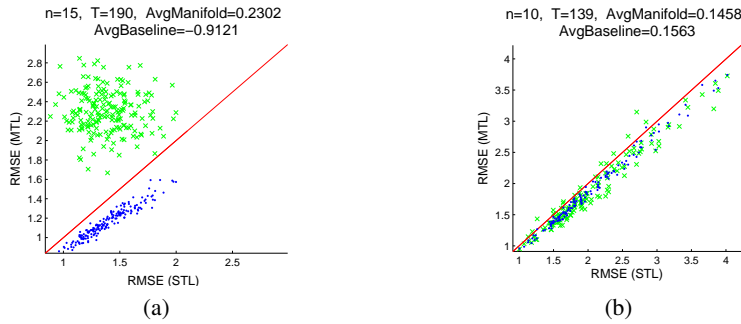

(a)  (b)

Figure 3: Taskwise performance on (a) computer and (b) school datasets.

features, giving us a total of 26 features. We again split the dataset into $75\%$ training and $25\%$ testing.

Similar to the synthetic dataset, hyperparameters of the baseline method and manifold method ($\gamma$ and $\lambda$) were tuned on a small validation dataset picked randomly from the training set. In the experiments, whenever we are required to use fewer number of examples, examples were chosen randomly. In such experiments, reported numbers were averaged over 10 runs for the statistical significance. Note that the fewer the examples, the higher the variance because of randomness. In order to see if learning tasks simultaneously helps, we did not consider the zero value while tuning the hyperparameters of MTL to avoid the reduction of MTL method to STL ones.

Figure 3(a) and Figure 3(b) shows the taskwise performance of the computer and school datasets respectively. We note that for the computer dataset, we perform significantly better than both STL and the baseline methods. The baseline method performs worse than the STL method, therefore giving a negative average performance improvement of $-0.9121$. We believe that this is because the tasks are related non-linearly. For the school dataset, we perform better than both STL and the baseline method though relative performance improvement is not as significant as in the computer dataset. On the school dataset, the baseline method has a mixed behavior relative to the STL method, performing good on some tasks while performing worse on others. In both of these datasets, we observe that our method does not cause the negative transfer i.e. causing a task to perform worse than the STL. Although we have not used anything in our problem formulation to avoid *negative transfer*, this observation is interesting. Note that almost all of the existing MTL methods suffer from the negative transfer phenomena. We emphasize that the baseline method has two parameters

that are very important, the regularization parameter and the $P$. In our experiments we found that the baseline method is very sensitive to both of these parameters. In order to have a fair and competitive comparison, we used the best value of these parameters, tuned on a small validation dataset picked randomly from the training set.

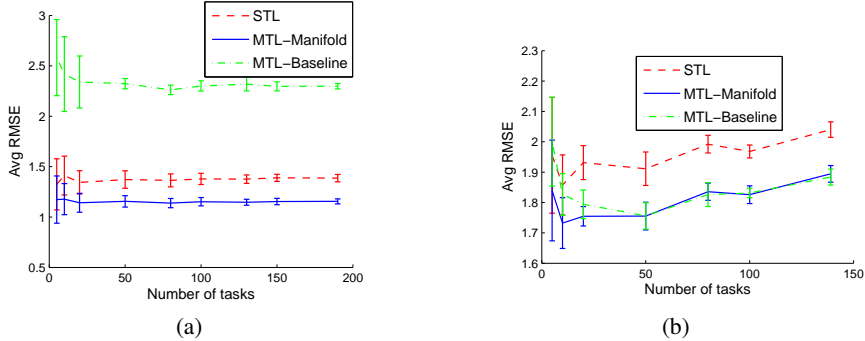

Figure 4: RMSE Vs number of tasks for (a) computer dataset (b) school dataset

Now we show the performance variation with respect to the number of training examples. Figure 2(b) shows the relative performance of the STL, MTL-baseline and MTL-Manifold for the school dataset. We outperform STL method significantly while we perform comparative to the baseline. Note that when the number of examples is relatively low, the baseline method outperforms our method because we do not have enough examples to estimate the parameters of the task which is used for the manifold construction. But as we increase the number of examples, we get better estimate of the parameters, hence better manifold regularization. For $n > 100$ we outperform the baseline method by a small amount. Variation of the performance with $n$ is not shown for the computer dataset because computer dataset has only 20 examples per task.

Performance variation with respect to the number of tasks for school and computer datasets is shown in Figure 4. We outperform STL method and the baseline method for the computer dataset while perform better/equal on the school dataset. These two plots indicate how the tasks are related in these two datasets. It suggests that tasks in school datasets are related linearly (Manifold and baseline methods have the same performance [4]) while tasks in the computer dataset are related non-linearly, which is why baseline method performs poor compared to the STL method. Both datasets exhibit the different behavior as we increase the number of tasks, though behavior relative to the STL method remains constant. This suggests that after a certain number of tasks, performance is not affected by adding more tasks. This is especially true for the computer dataset since it only has 13 features and only a few tasks are required to learn the task relatedness structure.

In summary, our method improves the performance over STL in all of these datasets (no negative transfer), while baseline method performs comparatively on the school dataset and performs worse on the computer dataset.

## 5 Conclusion

We have presented a novel method for multitask learning based on a natural and intuitive assumption about the task relatedness. We have used the manifold assumption to enforce the task relatedness which is a generalization of the previous notions of relatedness. Unlike many other previous approaches, our method does not require any other external information e.g. function/matrix other than the manifold assumption. We have performed experiments on synthetic and real datasets, and compared our results with the state-of-the-art method. We have shown that we outperform the baseline method in nearly all cases. We emphasize that unlike the baseline method, we improve over single task learning in almost all cases and do not encounter the negative transfer.

## Footnotes

[2]For a fair comparison, we use the code provided by the author, available at `http://ttic.uchicago.edu/˜argyriou/code/mtl_feat/mtl_feat.tar`.

[3]Available at `http://www.cmm.bristol.ac.uk/learning-training/multilevel-m-support/datasets.shtml`

[4]In the ideal case, the non-linear method should be able to discover the linear structure. But in practice, they might differ, especially when there are fewer number of tasks. This is the reason we perform equal on the school dataset when the number of tasks is high.

# References

[1] A. Argyriou, T. Evgeniou, and M. Pontil. Multi-task feature learning. In *NIPS '06*, 2006.

[2] A. Argyriou, T. Evgeniou, M. Pontil, A. Argyriou, T. Evgeniou, and M. Pontil. Convex multi-task feature learning. In *Machine Learning*. press, 2007.

[3] A. Argyriou, C. A. Micchelli, and M. Pontil. When is there a representer theorem? vector versus matrix regularizers. *J. Mach. Learn. Res.*, 10:2507–2529, 2009.

[4] A. Argyriou, C. A. Micchelli, M. Pontil, and Y. Ying. A spectral regularization framework for multi-task structure learning. In *NIPS '08*. 2008.

[5] B. Bakker and T. Heskes. Task clustering and gating for bayesian multitask learning. *JMLR*, 4:2003, 2003.

[6] M. Belkin and P. Niyogi. Laplacian eigenmaps for dimensionality reduction and data representation. *Neural Computation*, 15:1373–1396, 2002.

[7] M. Belkin, P. Niyogi, and V. Sindhwani. Manifold regularization: A geometric framework for learning from labeled and unlabeled examples. *J. Mach. Learn. Res.*, 7:2399–2434, 2006.

[8] R. Caruana. Multitask learning. In *Machine Learning*, pages 41–75, 1997.

[9] H. Daumé III. Bayesian multitask learning with latent hierarchies. In *Conference on Uncertainty in Artificial Intelligence '09*, Montreal, Canada, 2009.

[10] T. Evgeniou, C. A. Micchelli, and M. Pontil. Learning multiple tasks with kernel methods. *JMLR*, 6:615–637, 2005.

[11] S. Gerber, T. Tasdizen, and R. Whitaker. Dimensionality reduction and principal surfaces via kernel map manifolds. In *In Proceedings of the 2009 International Conference on Computer Vison (ICCV)*, 2009.

[12] T. Hastie. *Principal curves and surfaces*. PhD thesis, Stanford University, 1984.

[13] L. Jacob, F. Bach, and J.-P. Vert. Clustered multi-task learning: A convex formulation. In *NIPS '08*, 2008.

[14] P. J. Lenk, W. S. DeSarbo, P. E. Green, and M. R. Young. Hierarchical bayes conjoint analysis: Recovery of partworth heterogeneity from reduced experimental designs. *MARKETING SCIENCE*, 1996.

[15] Q. Liu, X. Liao, H. L. Carin, J. R. Stack, and L. Carin. Semisupervised multitask learning. *IEEE 2009*, 2009.

[16] C. A. Micchelli and M. Pontil. Regularized multi-task learning. In *KDD 2004*, pages 109–117, 2004.

[17] S. T. Roweis and L. K. Saul. Nonlinear dimensionality reduction by locally linear embedding. *Science*, 290(5500):2323–2326, December 2000.

[18] J. B. Tenenbaum, V. Silva, and J. C. Langford. A global geometric framework for nonlinear dimensionality reduction. *Science*, 290(5500):2319–2323, December 2000.

[19] S. Thrun and L. Pratt, editors. *Learning to learn*. Kluwer Academic Publishers, Norwell, MA, USA, 1998.

[20] K. Q. Weinberger, F. Sha, and L. K. Saul. Learning a kernel matrix for nonlinear dimensionality reduction. In *In ICML 2004*, pages 839–846. ACM Press, 2004.

[21] Y. Xue, X. Liao, L. Carin, and B. Krishnapuram. Multi-task learning for classification with dirichlet process priors. *J. Mach. Learn. Res.*, 8:35–63, 2007.

[22] K. Yu, V. Tresp, and A. Schwaighofer. Learning gaussian processes from multiple tasks. In *ICML '05*, 2005.

[23] J. Zhang, Z. Ghahramani, and Y. Yang. Flexible latent variable models for multi-task learning. *Mach. Learn.*, 73(3):221–242, 2008.

[24] J. Zhang, J. Zhang, Y. Yang, Z. Ghahramani, and Y. Yang. Learning multiple related tasks using latent independent component analysis. In *NIPS '05*, 2005.

